# Decoupling Sparsity and Smoothness in the Discrete Hierarchical Dirichlet Process

**Chong Wang**
Computer Science Department
Princeton University
chongw@cs.princeton.edu

**David M. Blei**
Computer Science Department
Princeton University
blei@cs.princeton.edu

## Abstract

We present a nonparametric hierarchical Bayesian model of document collections that decouples sparsity and smoothness in the component distributions (i.e., the "topics"). In the *sparse topic model* (sparseTM), each topic is represented by a bank of selector variables that determine which terms appear in the topic. Thus each topic is associated with a subset of the vocabulary, and topic smoothness is modeled on this subset. We develop an efficient Gibbs sampler for the sparseTM that includes a general-purpose method for sampling from a Dirichlet mixture with a combinatorial number of components. We demonstrate the sparseTM on four real-world datasets. Compared to traditional approaches, the empirical results will show that sparseTMs give better predictive performance with simpler inferred models.

## 1    Introduction

The hierarchical Dirichlet process (HDP) [1] has emerged as a powerful model for the unsupervised analysis of text. The HDP models documents as distributions over a collection of latent components, which are often called "topics" [2, 3]. Each word is assigned to a topic, and is drawn from a distribution over terms associated with that topic. The per-document distributions over topics represent systematic regularities of word use among the documents; the per-topic distributions over terms encode the randomness inherent in observations from the topics. The number of topics is unbounded.

Given a corpus of documents, analysis proceeds by approximating the posterior of the topics and topic proportions. This posterior bundles the two types of regularity. It is a probabilistic decomposition of the corpus into its systematic components, i.e., the distributions over topics associated with each document, and a representation of our uncertainty surrounding observations from each of those components, i.e., the topic distributions themselves. With this perspective, it is important to investigate how prior assumptions behind the HDP affect our inferences of these regularities.

In the HDP for document modeling, the topics are typically assumed drawn from an exchangeable Dirichlet, a Dirichlet for which the components of the vector parameter are equal to the same scalar parameter. As this scalar parameter approaches zero, it affects the Dirichlet in two ways. First, the resulting draws of random distributions will place their mass on only a few terms. That is, the resulting topics will be *sparse*. Second, given observations from such a Dirichlet, a small scalar parameter encodes increased confidence in the estimate from the observed counts. As the parameter approaches zero, the expectation of each per-term probability becomes closer to its empirical estimate. Thus, the expected distribution over terms becomes less *smooth*. The single scalar Dirichlet parameter affects both the sparsity of the topics and smoothness of the word probabilities within them.

When employing the exchangeable Dirichlet in an HDP, these distinct properties of the prior have consequences for both the global and local regularities captured by the model. Globally, posterior inference will prefer more topics because more sparse topics are needed to account for the observed

words of the collection. Locally, the per-topic distribution over terms will be less smooth—the posterior distribution has more confidence in its assessment of the per-topic word probabilities—and this results in less smooth document-specific predictive distributions.

The goal of this work is to decouple sparsity and smoothness in the HDP. With the *sparse topic model* (sparseTM), we can fit sparse topics with more smoothing. Rather than placing a prior for the entire vocabulary, we introduce a Bernoulli variable for each term and each topic to determine whether or not the term appears in the topic. Conditioned on these variables, each topic is represented by a multinomial distribution over its subset of the vocabulary, a sparse representation.

This prior smoothes only the relevant terms and thus the smoothness and sparsity are controlled through different hyper-parameters. As we will demonstrate, sparseTMs give better predictive performance with simpler models than traditional approaches.

## 2 Sparse Topic Models

Sparse topic models (sparseTMs) aim to separately control the number of terms in a topic, i.e., sparsity, and the probabilities of those words, i.e., smoothness. Recall that a topic is a pattern of word use, represented as a distribution over the fixed vocabulary of the collection. In order to decouple smoothness and sparsity, we define a topic on a random subset of the vocabulary (giving sparsity), and then model uncertainty of the probabilities on that subset (giving smoothness). For each topic, we introduce a Bernoulli variable for each term in the vocabulary that decides whether the term appears in the topic. Similar ideas of using Bernoulli variables to represent "on" and "off" have been seen in several other models, such as the noisy-OR model [4] and aspect Bernoulli model [5]. We can view this approach as a particular "spike and slab" prior [6] over Dirichlet distributions. The "spike" chooses the terms for the topic; the "slab" only smoothes those terms selected by the spike.

Assume the size of the vocabulary is $V$. A Dirichlet distribution over the topic is defined on a $V - 1$-simplex, i.e.,

$$\boldsymbol{\beta} \sim \text{Dirichlet}(\gamma \mathbf{1}), \tag{1}$$

where $\mathbf{1}$ is a $V$-length vector of 1s. In an sparseTM, the idea of imposing sparsity is to use Bernoulli variables to restrict the size of the simplex over which the Dirichlet distribution is defined. Let $\boldsymbol{b}$ be a $V$-length binary vector composed of $V$ Bernoulli variables. Thus $\boldsymbol{b}$ specifies a smaller simplex through the "on"s of its elements. The Dirichlet distribution over the restricted simplex is

$$\boldsymbol{\beta} \sim \text{Dirichlet}(\gamma \boldsymbol{b}), \tag{2}$$

which is a degenerate Dirichlet distribution over the sub-simplex specified by $\boldsymbol{b}$. In [7], Friedman and Singer use this type of distributions for language modeling.

Now we introduce the generative process of the sparseTM. The sparseTM is built on the hierarchical Dirichlet process for text, which we shorthand HDP-LDA. [1] In the Bayesian nonparametric setting the number of topics is not specified in advance or found by model comparison. Rather, it is inferred through posterior inference. The sparseTM assumes the following generative process:

1. For each topic $k \in \{1, 2, \ldots\}$, draw term selection proportion $\pi_k \sim \text{Beta}(r, s)$.
   - (a) For each term $v$, $1 \leq v \leq V$, draw term selector $b_{kv} \sim \text{Bernoulli}(\pi_k)$.
   - (b) Let $b_{V+1} = \mathbf{1}[\sum_{v=1}^{V} b_{kv} = 0]$ and $\boldsymbol{b}_k = [b_{kv}]_{v=1}^{V+1}$.
     Draw topic distribution $\boldsymbol{\beta}_k \sim \text{Dirichlet}(\gamma \boldsymbol{b}_k)$.
2. Draw stick lengths $\boldsymbol{\alpha} \sim \text{GEM}(\lambda)$, which are the global topic proportions.
3. For document $d$:
   - (a) Draw per-document topic proportions $\boldsymbol{\theta}_d \sim \text{DP}(\tau, \boldsymbol{\alpha})$.
   - (b) For the $i$th word:
     - i. Draw topic assignment $z_{di} \sim \text{Mult}(\boldsymbol{\theta}_d)$.
     - ii. Draw word $w_{di} \sim \text{Mult}(\beta_{z_{di}})$

Figure 1 illustrates the sparseTM as a graphical model.

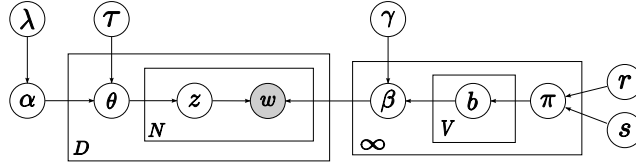

Figure 1: A graphical model representation for sparseTMs.

The distinguishing feature of the sparseTM is step 1, which generates the latent topics in such a way that decouples sparsity and smoothness. For each topic $k$ there is a corresponding Beta random variable $\pi_k$ and a set of Bernoulli variables $b_{kv}$s, one for each term in the vocabulary. Define the sparsity of the topic as

$$\text{sparsity}_k \triangleq 1 - \sum_{v=1}^{V} b_{kv}/V. \tag{3}$$

This is the proportion of zeros in its bank of Bernoulli random variables. Conditioned on the Bernoulli parameter $\pi_k$, the expectation of the sparsity is

$$\mathbb{E}\left[\text{sparsity}_k | \pi_k\right] = 1 - \pi_k. \tag{4}$$

The conditional distribution of the topic $\boldsymbol{\beta}_k$ given the vocabulary subset $\boldsymbol{b}_k$ is Dirichlet($\gamma \boldsymbol{b}_k$). Thus, topic $k$ is represented by those terms with non-zero $b_{kv}$s, and the smoothing is only enforced over these terms through hyperparameter $\gamma$. Sparsity, which is determined by the pattern of ones in $\boldsymbol{b}_k$, is controlled by the Bernoulli parameter. Smoothing and sparsity are decoupled.

One nuance is that we introduce $b_{V+1} = 1[\sum_{v=1}^{V} b_{kv} = 0]$. The reason is that when $\boldsymbol{b}_{k,1:V} = \boldsymbol{0}$, Dirichlet($\gamma \boldsymbol{b}_{k,1:V}$) is not well defined. The term $b_{V+1}$ extends the vocabulary to $V+1$ terms, where the $V+1$th term *never* appears in the documents. Thus, Dirichlet($\gamma \boldsymbol{b}_{k,1:V+1}$) is always well defined.

We next compute the marginal distribution of $\boldsymbol{\beta}_k$, after integrating out Bernoullis $\boldsymbol{b}_k$ and their parameter $\pi_k$:

$$p(\boldsymbol{\beta}_k \,|\, \gamma, r, s) = \int d\pi_k \, p(\boldsymbol{\beta}_k \,|\, \gamma, \pi_k) p(\pi_k | r, s)$$

$$= \sum_{\boldsymbol{b}_k} p(\boldsymbol{\beta}_k \,|\, \gamma, \boldsymbol{b}_k) \int d\pi_k \, p(\boldsymbol{b}_k | \pi_k) p(\pi_k | r, s).$$

We see that $p(\boldsymbol{\beta}_k \,|\, \gamma, r, s)$ and $p(\boldsymbol{\beta}_k \,|\, \gamma, \pi_k)$ are mixtures of Dirichlet distributions, where the mixture components are defined over simplices of different dimensions. In total, there are $2^V$ components; each configuration of Bernoulli variables $\boldsymbol{b}_k$ specifies one particular component. In posterior inference we will need to sample from this distribution. Sampling from such a mixture is difficult in general, due to the combinatorial sum. In the supplement, we present an efficient procedure to overcome this issue. This is the central computational challenge for the sparseTM.

Step 2 and 3 mimic the generative process of HDP-LDA [1]. The stick lengths $\alpha$ come from a Griffiths, Engen, and McCloskey (GEM) distribution [8], which is drawn using the stick-breaking construction [9],

$$\eta_k \sim \text{Beta}(1, \lambda),$$
$$\alpha_k = \eta_k \prod_{j=1}^{k-1}(1 - \eta_j), \quad k \in \{1, 2, \dots\}.$$

Note that $\sum_k \alpha_k = 1$ almost surely. The stick lengths are used as a base measure in the Dirichlet process prior on the per-document topic proportions, $\theta_d \sim \text{DP}(\tau, \boldsymbol{\alpha})$. Finally, the generative process for the topic assignments $\boldsymbol{z}$ and observed words $\boldsymbol{w}$ is straightforward.

## 3 Approximate posterior inference using collapsed Gibbs sampling

Since the posterior inference is intractable in sparseTMs, we turn to a collapsed Gibbs sampling algorithm for posterior inference. In order to do so, we integrate out topic proportions $\boldsymbol{\theta}$, topic distributions $\boldsymbol{\beta}$ and term selectors $\boldsymbol{b}$ analytically. The latent variables needed by the sampling algorithm

are stick lengths $\boldsymbol{\alpha}$, Bernoulli parameter $\boldsymbol{\pi}$ and topic assignment $\boldsymbol{z}$. We fix the hyperparameter $s$ equal to 1.

To sample $\boldsymbol{\alpha}$ and topic assignments $\boldsymbol{z}$, we use the direct-assignment method, which is based on an analogy to the Chinese restaurant franchise (CRF) [1]. To apply direct assignment sampling, an auxiliary table count random variable $\boldsymbol{m}$ is introduced. In the CRF setting, we use the following notation. The number of customers in restaurant $d$ (document) eating dish $k$ (topic) is denoted $n_{dk}$, and $n_{d\cdot}$ denotes the number of customers in restaurant $d$. The number of tables in restaurant $d$ serving dish $k$ is denoted $m_{dk}$, $m_{d\cdot}$ denotes the number of tables in restaurant $d$, $m_{\cdot k}$ denotes the number of tables serving dish $k$, and $m_{\cdot\cdot}$ denotes the total number of tables occupied. (Marginal counts are represented with dots.) Let $K$ be the current number of topics. The function $n_k^{(v)}$ denotes the number of times that term $v$ has been assigned to topic $k$, while $n_k^{(\cdot)}$ denotes the number of times that all the terms have been assigned to topic $k$. Index $u$ is used to indicate the new topic in the sampling process. Note that direct assignment sampling of $\boldsymbol{\alpha}$ and $\boldsymbol{z}$ is conditioned on $\boldsymbol{\pi}$.

The crux for sampling stick lengths $\boldsymbol{\alpha}$ and topic assignments $\boldsymbol{z}$ (conditioned on $\boldsymbol{\pi}$) is to compute the conditional density of $w_{di}$ under the topic component $k$ given all data items except $w_{di}$ as,

$$f_k^{-w_{di}}(w_{di} = v|\pi_k) \triangleq p(w_{di} = v|\{w_{d'i'}, z_{d'i'} : z_{d'i'} = k, d'i' \neq di\}, \pi_k). \tag{5}$$

The derivation of equations for computing this conditional density is detailed in the supplement.[2] We summarize our findings as follows. Let $\mathcal{V} \triangleq \{1, \dots, V\}$ be the set of vocabulary terms, $B_k \triangleq \{v : n_{k,-di}^{(v)} > 0, v \in \mathcal{V}\}$ be the set of terms that have word assignments in topic $k$ after excluding $w_{di}$ and $|B_k|$ be its cardinality. Let's assume that $B_k$ is not an empty set.[3] We have the following,

$$f_k^{-w_{di}}(w_{di} = v|\pi_k) \propto \begin{cases} (n_{k,-di}^{(v)} + \gamma)\mathbb{E}\left[g_{B_k}(X)|\pi_k\right] & \text{if } v \in B_k \\ \gamma\pi_k\mathbb{E}\left[g_{\bar{B}_k}(\bar{X})|\pi_k\right] & \text{otherwise.} \end{cases}, \tag{6}$$

where

$$g_{B_k}(x) = \frac{\Gamma((|B_k| + x)\gamma)}{\Gamma(n_{k,-di}^{(\cdot)} + 1 + (|B_k| + x)\gamma)},$$
$$X \mid \pi_k \sim \text{Binomial}(V - |B_k|, \pi_k),$$
$$\bar{X} \mid \pi_k \sim \text{Binomial}(V - |\bar{B}_k|, \pi_k), \tag{7}$$

and where $\bar{B}_k = B_k \cup \{v\}$. Further note $\Gamma(\cdot)$ is the Gamma function and $n_{k,-di}^{(v)}$ describes the corresponding count excluding word $w_{di}$. In the supplement, we also show that $\mathbb{E}\left[g_{B_k}(X)|\pi_k\right] > \pi_k\mathbb{E}\left[g_{\bar{B}_k}(\bar{X})|\pi_k\right]$. The central difference between the algorithms for HDP-LDA and the sparseTM is conditional probability in Equation 6 which depends on the selector variables and selector proportions.

We now describe how we sample stick lengths $\boldsymbol{\alpha}$ and topic assignments $\boldsymbol{z}$. This is similar to the sampling procedure for HDP-LDA [1].

**Sampling stick lengths $\boldsymbol{\alpha}$.** Although $\boldsymbol{\alpha}$ is an infinite-length vector, the number of topics $K$ is finite at every point in the sampling process. Sampling $\boldsymbol{\alpha}$ can be replaced by sampling $\boldsymbol{\alpha} \triangleq [\alpha_1, \dots, \alpha_K, \alpha_u]$ [1]. That is,

$$\boldsymbol{\alpha} \mid \boldsymbol{m} \sim \text{Dirichlet}(m_{\cdot 1}, \dots, m_{\cdot K}, \lambda). \tag{8}$$

**Sampling topic assignments $\boldsymbol{z}$.** This is similar to the sampling approach for HDP-LDA [1] as well. Using the conditional density $f$ defined Equation 5 and 6, we have

$$p(z_{di} = k|\boldsymbol{z}_{-di}, \boldsymbol{m}, \boldsymbol{\alpha}, \pi_k) \propto \begin{cases} (n_{dk,-di} + \tau\alpha_k)f_k^{-w_{di}}(w_{di}|\pi_k) & \text{if } k \text{ previously used,} \\ \tau\alpha_u f_u^{-w_{di}}(w_{di}|\pi_u) & k = u. \end{cases} \tag{9}$$

If a new topic $k_{new}$ is sampled, then sample $\varkappa \sim \text{Beta}(1, \lambda)$, and let $\alpha_{k_{new}} = \varkappa\alpha_u$ and $\alpha_{u^{new}} = (1 - \varkappa)\alpha_u$.

**Sampling Bernoulli parameter $\pi$.** To sample $\pi_k$, we use $\boldsymbol{b}_k$ as an auxiliary variable. Note that $\boldsymbol{b}_k$ was integrated out earlier. Recall $B_k$ is the set of terms that have word assignments in topic $k$. (This time, we don't need to exclude certain words since we are sampling $\pi$.) Let $A_k = \{v : b_{kv} = 1, v \in \mathcal{V}\}$ be the set of the indices of $b_k$ that are "on", the joint conditional distribution of $\pi_k$ and $\boldsymbol{b}_k$ is

$$p(\pi_k, \boldsymbol{b}_k|\text{rest}) \propto p(\boldsymbol{b}_k|\pi_k)p(\pi_k|r)p(\{w_{di} : z_{di} = k\}|\boldsymbol{b}_k, \{z_{di} : z_{di} = k\})$$

$$= p(\boldsymbol{b}_k|\pi_k)p(\pi_k|r) \int d\boldsymbol{\beta}_k \, p(\{w_{di} : z_{di} = k\}|\boldsymbol{\beta}_k, \{z_{di} : z_{di} = k\})p(\boldsymbol{\beta}_k|\boldsymbol{b}_k)$$

$$= p(\boldsymbol{b}_k|\pi_k)p(\pi_k|r)\frac{1_{B_k \subset A_k}\Gamma(|A_k|\gamma)\prod_{v \in A_k}\Gamma(n_k^{(v)} + \gamma)}{\Gamma^{|A_k|}(\gamma)\Gamma(n_k^{(\cdot)} + |A_k|\gamma)}$$

$$= p(\boldsymbol{b}_k|\pi_k)p(\pi_k|r)\frac{1_{B_k \subset A_k}\Gamma(|A_k|\gamma)\prod_{v \in B_k}\Gamma(n_k^{(v)} + \gamma)}{\Gamma^{|B_k|}(\gamma)\Gamma(n_k^{(\cdot)} + |A_k|\gamma)}$$

$$\propto \prod_v p(b_{kv}|\pi_k)p(\pi_k|r)\frac{1_{B_k \subset A_k}\Gamma(|A_k|\gamma)}{\Gamma(n_k^{(\cdot)} + |A_k|\gamma)}, \tag{10}$$

where $1_{B_k \subset A_k}$ is an indicator function and $|A_k| = \sum_v b_{kv}$. This follows because if $A_k$ is not a super set of $B_k$, there must be a term, say $v$ in $B_k$ but not in $A_k$, causing $\beta_{kv} = 0$, a.s., and then $p(\{w_{di} : (d, i) \in Z_k\}|\boldsymbol{\beta}_k, \{z_{di} : (d, i) \in Z_k\}) = 0$ a.s.. Using this joint conditional distribution[4], we iteratively sample $\boldsymbol{b}_k$ conditioned on $\pi_k$ and $\pi_k$ conditioned on $\boldsymbol{b}_k$ to ultimately obtain a sample from $\pi_k$.

**Others.** Sampling the table counts $\boldsymbol{m}$ is exactly the same as for the HDP [1], so we omit the details here. In addition, we can sample the hyper-parameters $\lambda$, $\tau$ and $\gamma$. For the concentration parameters $\lambda$ and $\tau$ in both HDP-LDA and sparseTMs, we use previously developed approaches for Gamma priors [1, 10]. For the Dirichlet hyper-parameter $\gamma$, we use Metropolis-Hastings.

Finally, with any single sample we can estimate topic distributions $\boldsymbol{\beta}$ from the value topic assignments $\boldsymbol{z}$ and term selector $\boldsymbol{b}$ by

$$\hat{\beta}_{k,v} = \frac{n_k^{(v)} + b_{k,v}\gamma}{n_k^{(\cdot)} + \sum_v b_{kv}\gamma}, \tag{11}$$

where we can smooth only those terms that are chosen to be in the topics. Note that we can obtain the samples of $\boldsymbol{b}$ when sampling the Bernoulli parameter $\boldsymbol{\pi}$.

## 4 Experiments

In this section, we studied the performance of the sparseTM on four datasets and demonstrated how sparseTM decouples the smoothness and sparsity in the HDP.[5] We placed Gamma$(1, 1)$ priors over the hyper-parameters $\lambda$ and $\tau$. The sparsity proportion prior was a uniform Beta, i.e., $r = s = 1$. For hyper-parameter $\gamma$, we use Metropolis-Hastings sampling method using symmetric Gaussian proposal with variance 1.0. A disadvantage of sparseTM is that its running speed is about 4-5 times slower than the HDP-LDA.

### 4.1 Datasets

The four datasets we use in the experiments are:

1. The *arXiv* data set contains 2500 (randomly sampled) online research abstracts (http://arxiv.org). It has 2873 unique terms, around 128K observed words and an average of 36 unique terms per document.

2. The *Nematode Biology* data set contains 2500 (randomly sampled) research abstracts (http://elegans.swmed.edu/wli/cgcbib). It has 2944 unique terms, around 179K observed words and an average of 52 unique terms per document.
3. The *NIPS* data set contains the NIPS articles published between 1988-1999 (http://www.cs.utoronto.ca/∼sroweis/nips). It has 5005 unique terms and around 403K observed words. We randomly sample 20% of the words for each paper and this leads to an average of 150 unique terms per document.
4. The *Conf. abstracts* set data contains abstracts (including papers and posters) from six international conferences: CIKM, ICML, KDD, NIPS, SIGIR and WWW (http://www.cs.princeton.edu/∼chongw/data/6conf.tgz). It has 3733 unique terms, around 173K observed words and an average of 46 unique terms per document. The data are from 2005-2008.

For all data, stop words and words occurring fewer than 10 times were removed.

## 4.2 Performance evaluation and model examinations

We studied the predictive performance of the sparseTM compared to HDP-LDA. On the training documents our Gibbs sampler uses the first 2000 steps as burn-in, and we record the following 100 samples as samples from the posterior. Conditioned on these samples, we run the Gibbs sampler for test documents to estimate the predictive quantities of interest. We use 5-fold cross validation.

We study two predictive quantities. First, we examine overall predictive power with the *predictive perplexity* of the test set given the training set. (This is a metric from the natural language literature.) The predictive perplexity is

$$\text{perplexity}_{\text{pw}} = \exp\left\{ -\frac{\sum_{d \in D_{\text{test}}} \log p(\mathbf{w}_d | D_{\text{train}})}{\sum_{d \in D_{test}} N_d} \right\}.$$

Lower perplexity is better.

Second, we compute *model complexity*. Nonparametric Bayesian methods are often used to sidestep model selection and integrate over all instances (and all complexities) of a model at hand (e.g., the number of clusters). The model, though hidden and random, still lurks in the background. Here we study its posterior distribution with the desideratum that between two equally good predictive distributions, a simpler model—or a posterior peaked at a simpler model—is preferred.

To capture model complexity we first define the complexity of topic. Recall that each Gibbs sample contains a topic assignment $z$ for every observed word in the corpus (see Equation 9). The topic complexity is the number of unique terms that have at least one word assigned to the topic. This can be expressed as a sum of indicators,

$$\text{complexity}_k = \sum_d 1\left[\left(\sum_n 1[z_{d,n} = k]\right) > 0\right],$$

where recall that $z_{d,n}$ is the topic assignment for the $n$th word in document $d$. Note a topic with no words assigned to it has complexity zero. For a particular Gibbs sample, the model complexity is the sum of the topic complexities and the number of topics. Loosely, this is the number of free parameters in the "model" that the nonparametric Bayesian method has selected, which is

$$\text{complexity} = \#\text{topics} + \sum_k \text{complexity}_k. \tag{12}$$

We performed posterior inference with the sparseTM and HDP-LDA, computing predictive perplexity and average model complexity with 5-fold cross validation. Figure 2 illustrates the results.

**Perplexity versus Complexity.** Figure 2 (first row) shows the model complexity versus predictive perplexity for each fold: Red circles represent sparseTM, blue squares represent HDP-LDA, and the dashed line connecting a red circle and blue square indicates the that the two are from the same fold. These results shows that the sparseTM achieves better perplexity than HDP-LDA, and at *simpler* models. (To see this, notice that all the connecting lines going from HDP-LDA to sparseTM point down and to the left.)

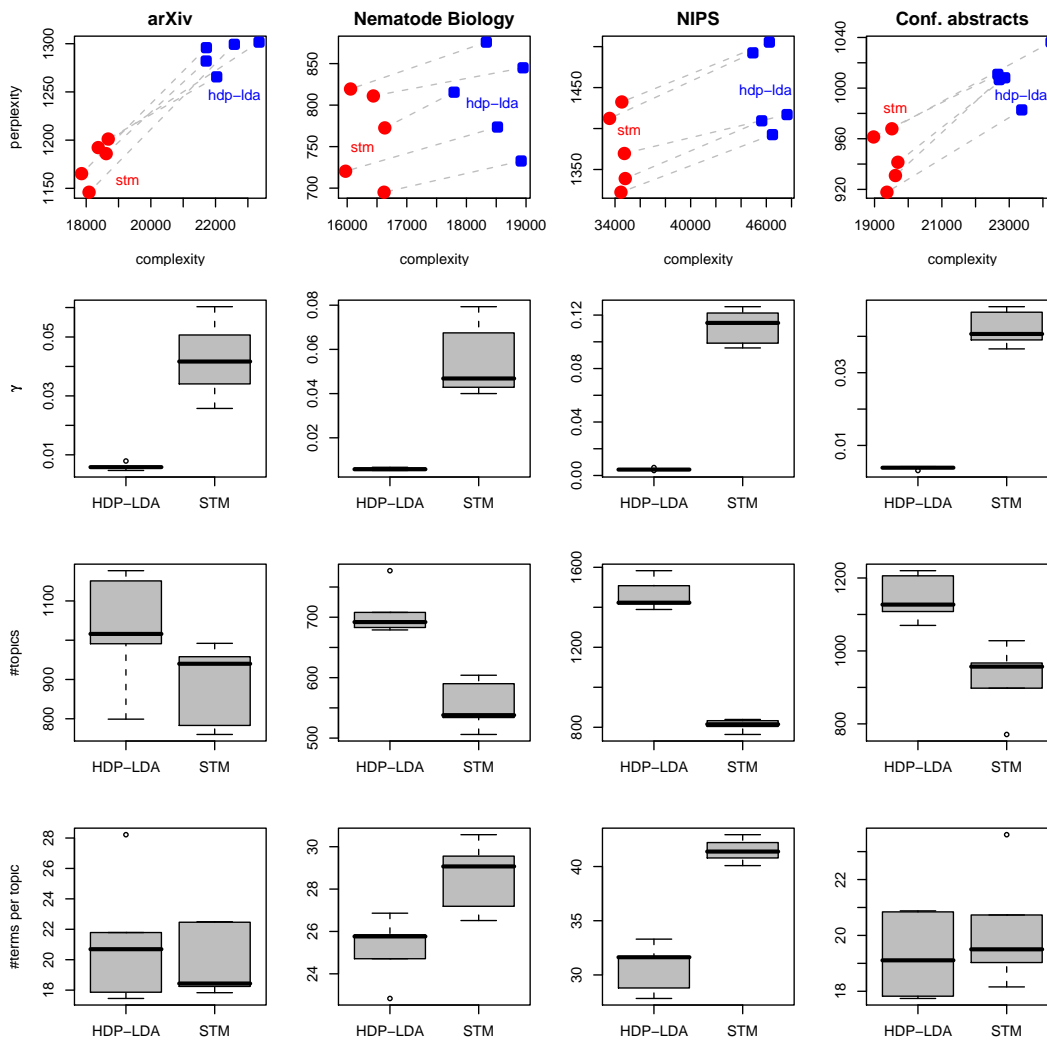

Figure 2: Experimental results for sparseTM (shortened as STM in this figure) and HDP-LDA on four datasets. **First row**. The scatter plots of model complexity versus predictive perplexity for 5-fold cross validation: Red circles represent the results from sparseTM, blue squares represent the results from HDP-LDA and the dashed lines connect results from the same fold. **Second row**. Box plots of the hyperparameter $\gamma$ values. **Third row**. Box plots of the number of topics. **Fourth row**. Box plots of the number of terms per topic.

**Hyperparameter $\gamma$, number of topics and number of terms per topic.**  Figure 2 (from the second to fourth rows) shows the Dirichlet parameter $\gamma$ and posterior number of topics for HDP-LDA and sparseTM. HDP-LDA tends to have a very small $\gamma$ in order to attain a reasonable number of topics, but this leads to less smooth distributions. In contrast, sparseTM allows a larger $\gamma$ and selects more smoothing, even with a smaller number of topics. The numbers of terms per topic for two models don't have a consistent trend, but they don't differ too much either.

**Example topics.**  For the NIPS data set, we provide some example topics (with top 15 terms) discovered by HDP-LDA and sparseTM in Table 1. Accidentally, we found that HDP-LDA seems to produce more noisy topics, such as, those shown in Table 2.

| sparseTM | HDP-LDA | sparseTM | HDP-LDA |  | Example "noise topics" |  |
|---|---|---|---|---|---|---|
| support | svm | belief | variational |  | epsilon | resulting |
| vector | vector | networks | networks |  | stream | mation |
| svm | support | inference | jordan |  | direct | inferred |
| kernel | machines | lower | parameters |  | development | transfer |
| machines | kernel | bound | inference |  | behaviour | depicted |
| margin | svms | variational | bound |  | motor | global |
| training | decision | jordan | belief |  | corner | submitted |
| vapnik | http | graphical | distributions |  | carried | inter |
| solution | digit | exact | approximation |  | applications | applicable |
| examples | machine | field | lower |  | mixture | replicated |
| space | diagonal | probabilistic | methods |  | served | refers |
| sv | regression | approximate | quadratic |  | specification | searching |
| note | sparse | conditional | field |  | modest | operates |
| kernels | optimization | variables | distribution |  | tension | vertical |
| svms | misclassification | models | intractable |  | matter | class |

Table 1: Similar topics discovered.　　　　Table 2: "Noise" topics in HDP-LDA.

## 5 Discussion

These results illuminate the issue with a single parameter controlling both sparsity and smoothing. In the Gibbs sampler, if the HDP-LDA posterior requires more topics to explain the data, it will reduce the value of $\gamma$ to accommodate for the increased (necessary) sparseness. This smaller $\gamma$, however, leads to less smooth topics that are less robust to "noise", i.e., infrequent words that might populate a topic. The process is circular: To explain the noisy words, the Gibbs sampler might invoke new topics still, thereby further reducing the hyperparameter. As a result of this interplay, HDP-LDA settles on more topics and a smaller $\gamma$. Ultimately, the fit to held out data suffers.

For the sparseTM, however, more topics can be used to explain the data by using the sparsity control gained from the "spike" component of the prior. The hyperparameter $\gamma$ is controlled separately. Thus the smoothing effect is retained, and held out performance is better.

**Acknowledgements.** We thank anonymous reviewers for insightful suggestions. David M. Blei is supported by ONR 175-6343, NSF CAREER 0745520, and grants from Google and Microsoft.

## Footnotes

[1]This acronym comes from the fact that the HDP for text is akin to a nonparametric Bayesian version of latent Dirichlet allocation (LDA).

[2]Note we integrate out $\boldsymbol{\beta}_k$ and $\boldsymbol{b}_k$. Another sampling strategy is to sample $\boldsymbol{b}$ (by integrating out $\boldsymbol{\pi}$) and the Gibbs sampler is much easier to derive. However, conditioned on $\boldsymbol{b}$, sampling $\boldsymbol{z}$ will be constrained to a smaller set of topics (specified by the values of $\boldsymbol{b}$), which slows down convergence of the sampler.

[3]In the supplement, we show that if $B_k$ is an empty set, the result is trivial.

[4]In our experiments, we used the algorithm described in the main text to sample $\pi$. We note that an improved algorithm might be achieved by modeling the joint conditional distribution of $\pi_k$ and $\sum_v b_{kv}$ instead, i.e., $p(\pi_k, \sum_v b_{kv}|\text{rest})$, since sampling $\pi_k$ only depends on $\sum_v b_{kv}$.

[5]Other experiments, which we don't report here, also showed that the finite version of sparseTM outperforms LDA with the same number of topics.

## References

[1] Teh, Y. W., M. I. Jordan, M. J. Beal, et al. Hierarchical Dirichlet processes. *Journal of the American Statistical Association*, 101(476):1566–1581, 2006.

[2] Blei, D., A. Ng, M. Jordan. Latent Dirichlet allocation. *J. Mach. Learn. Res.*, 3:993–1022, 2003.

[3] Griffths, T., M. Steyvers. Probabilistic topic models. In *Latent Semantic Analysis: A Road to Meaning*. 2006.

[4] Saund, E. A multiple cause mixture model for unsupervised learning. *Neural Comput.*, 7(1):51–71, 1995.

[5] Kabán, A., E. Bingham, T. Hirsimäki. Learning to read between the lines: The aspect Bernoulli model. In *SDM*. 2004.

[6] Ishwaran, H., J. S. Rao. Spike and slab variable selection: Frequentist and Bayesian strategies. *The Annals of Statistics*, 33(2):730–773, 2005.

[7] Friedman, N., Y. Singer. Efficient Bayesian parameter estimation in large discrete domains. In *NIPS*. 1999.

[8] Pitman, J. Poisson–Dirichlet and GEM invariant distributions for split-and-merge transformations of an interval partition. *Comb. Probab. Comput.*, 11(5):501–514, 2002.

[9] Sethuraman, J. A constructive definition of Dirichlet priors. *Statistica Sinica*, 4:639–650, 1994.

[10] Escobar, M. D., M. West. Bayesian density estimation and inference using mixtures. *Journal of the American Statistical Association*, 90:577–588, 1995.

